# Reinforcement Learning for Trading

**John Moody and Matthew Saffell***
Oregon Graduate Institute, CSE Dept.
P.O. Box 91000, Portland, OR 97291–1000
{moody, saffell}@cse.ogi.edu

## Abstract

We propose to train trading systems by optimizing financial objective functions via reinforcement learning. The performance functions that we consider are profit or wealth, the Sharpe ratio and our recently proposed *differential Sharpe ratio* for online learning. In Moody & Wu (1997), we presented empirical results that demonstrate the advantages of reinforcement learning relative to supervised learning. Here we extend our previous work to compare Q-Learning to our Recurrent Reinforcement Learning (RRL) algorithm. We provide new simulation results that demonstrate the presence of predictability in the monthly S&P 500 Stock Index for the 25 year period 1970 through 1994, as well as a sensitivity analysis that provides economic insight into the trader's structure.

## 1 Introduction: Reinforcement Learning for Trading

The investor's or trader's ultimate goal is to optimize some relevant measure of trading system performance, such as profit, economic utility or risk-adjusted return. In this paper, we propose to use recurrent reinforcement learning to directly optimize such trading system performance functions, and we compare two different reinforcement learning methods. The first, Recurrent Reinforcement Learning, uses immediate rewards to train the trading systems, while the second (Q-Learning (Watkins 1989)) approximates discounted future rewards. These methodologies can be applied to optimizing systems designed to trade a single security or to trade portfolios. In addition, we propose a novel value function for risk-adjusted return that enables learning to be done online: the *differential Sharpe ratio*.

Trading system profits depend upon sequences of interdependent decisions, and are thus path-dependent. Optimal trading decisions when the effects of transactions costs, market impact and taxes are included require knowledge of the current system state. In Moody, Wu, Liao & Saffell (1998), we demonstrate that reinforcement learning provides a more elegant and effective means for training trading systems when transaction costs are included, than do more standard supervised approaches.

Though much theoretical progress has been made in recent years in the area of reinforcement learning, there have been relatively few successful, practical applications of the techniques. Notable examples include Neuro-gammon (Tesauro 1989), the asset trader of Neuneier (1996), an elevator scheduler (Crites & Barto 1996) and a space-shuttle payload scheduler (Zhang & Dietterich 1996).

In this paper we present results for reinforcement learning trading systems that outperform the S&P 500 Stock Index over a 25-year test period, thus demonstrating the presence of predictable structure in US stock prices. The reinforcement learning algorithms compared here include our new recurrent reinforcement learning (RRL) method (Moody & Wu 1997, Moody et al. 1998) and Q-Learning (Watkins 1989).

## 2 Trading Systems and Financial Performance Functions

### 2.1 Structure, Profit and Wealth for Trading Systems

We consider performance functions for systems that trade a single [1] security with price series $z_t$. The trader is assumed to take only long, neutral or short positions $F_t \in \{-1, 0, 1\}$ of constant magnitude. The constant magnitude assumption can be easily relaxed to enable better risk control. The position $F_t$ is established or maintained at the end of each time interval $t$, and is re-assessed at the end of period $t + 1$. A trade is thus possible at the end of each time period, although nonzero trading costs will discourage excessive trading. A trading system return $R_t$ is realized at the end of the time interval $(t - 1, t]$ and includes the profit or loss resulting from the position $F_{t-1}$ held during that interval and any transaction cost incurred at time $t$ due to a difference in the positions $F_{t-1}$ and $F_t$.

In order to properly incorporate the effects of transactions costs, market impact and taxes in a trader's decision making, the trader must have internal state information and must therefore be recurrent. An example of a single asset trading system that takes into account transactions costs and market impact has following decision function: $F_t = F(\theta_t; F_{t-1}, I_t)$ with $I_t = \{z_t, z_{t-1}, z_{t-2}, \ldots; y_t, y_{t-1}, y_{t-2}, \ldots\}$ where $\theta_t$ denotes the (learned) system parameters at time $t$ and $I_t$ denotes the information set at time $t$, which includes present and past values of the price series $z_t$ and an arbitrary number of other external variables denoted $y_t$.

Trading systems can be optimized by maximizing performance functions $U()$ such as profit, wealth, utility functions of wealth or performance ratios like the Sharpe ratio. The simplest and most natural performance function for a risk-insensitive trader is profit. The transactions cost rate is denoted $\delta$.

**Additive profits** are appropriate to consider if each trade is for a fixed number of shares or contracts of security $z_t$. This is often the case, for example, when trading small futures accounts or when trading standard US$ FX contracts in dollar-denominated foreign currencies. With the definitions $r_t = z_t - z_{t-1}$ and $r_t^f = z_t^f - z_{t-1}^f$ for the price returns of a risky (traded) asset and a risk-free asset (like T-Bills) respectively, the additive profit accumulated over $T$ time periods with trading position size $\mu > 0$ is then defined as:

$$P_T = \sum_{t=1}^{T} R_t = \mu \sum_{t=1}^{T} \left\{ r_t^f + F_{t-1}(r_t - r_t^f) - \delta |F_t - F_{t-1}| \right\} \tag{1}$$

with $P_0 = 0$ and typically $F_T = F_0 = 0$. Equation (1) holds for continuous quantities also. The wealth is defined as $W_T = W_0 + P_T$.

**Multiplicative profits** are appropriate when a fixed fraction of accumulated wealth $\nu > 0$ is invested in each long or short trade. Here, $r_t = (z_t/z_{t-1} - 1)$ and $r_t^f = (z_t^f/z_{t-1}^f - 1)$. If no short sales are allowed and the leverage factor is set fixed at $\nu = 1$, the wealth at time T is:

$$W_T = W_0 \prod_{t=1}^{T} \{1 + R_t\} = W_0 \prod_{t=1}^{T} \left\{ 1 + (1 - F_{t-1})r_t^f + F_{t-1}r_t \right\} \{1 - \delta|F_t - F_{t-1}|\}. \quad (2)$$

## 2.2 The *Differential* Sharpe Ratio for On-line Learning

Rather than maximizing profits, most modern fund managers attempt to maximize risk-adjusted return as advocated by Modern Portfolio Theory. The Sharpe ratio is the most widely-used measure of risk-adjusted return (Sharpe 1966). Denoting as before the trading system returns for period $t$ (including transactions costs) as $R_t$, the Sharpe ratio is defined to be

$$S_T = \frac{\text{Average}(R_t)}{\text{Standard Deviation}(R_t)} \quad (3)$$

where the average and standard deviation are estimated for periods $t = \{1, \ldots, T\}$.

Proper on-line learning requires that we compute the influence on the Sharpe ratio of the return at time $t$. To accomplish this, we have derived a new objective function called the *differential Sharpe ratio* for on-line optimization of trading system performance (Moody *et al.* 1998). It is obtained by considering exponential moving averages of the returns and standard deviation of returns in (3), and expanding to first order in the decay rate $\eta$: $S_t \approx S_{t-1} + \eta \frac{dS_t}{d\eta}|_{\eta=0} + O(\eta^2)$. Noting that only the first order term in this expansion depends upon the return $R_t$ at time $t$, we define the *differential Sharpe ratio* as:

$$D_t \equiv \frac{dS_t}{d\eta} = \frac{B_{t-1}\Delta A_t - \frac{1}{2}A_{t-1}\Delta B_t}{(B_{t-1} - A_{t-1}^2)^{3/2}} \quad . \quad (4)$$

where the quantities $A_t$ and $B_t$ are exponential moving estimates of the first and second moments of $R_t$:

$$\begin{aligned} A_t &= A_{t-1} + \eta\Delta A_t = A_{t-1} + \eta(R_t - A_{t-1}) \\ B_t &= B_{t-1} + \eta\Delta B_t = B_{t-1} + \eta(R_t^2 - B_{t-1}) \quad . \end{aligned} \quad (5)$$

Treating $A_{t-1}$ and $B_{t-1}$ as numerical constants, note that $\eta$ in the update equations controls the magnitude of the influence of the return $R_t$ on the Sharpe ratio $S_t$. Hence, the *differential Sharpe ratio* represents the influence of the trading return $R_t$ realized at time $t$ on $S_t$.

## 3 Reinforcement Learning for Trading Systems

The goal in using reinforcement learning to adjust the parameters of a system is to maximize the expected payoff or reward that is generated due to the actions of the system. This is accomplished through trial and error exploration of the environment. The system receives a reinforcement signal from its environment (a

*reward*) that provides information on whether its actions are good or bad. The performance function at time $T$ can be expressed as a function of the sequence of trading returns $U_T = U(R_1, R_2, \ldots, R_T)$.

Given a trading system model $F_t(\theta)$, the goal is to adjust the parameters $\theta$ in order to maximize $U_T$. This maximization for a complete sequence of $T$ trades can be done off-line using dynamic programming or batch versions of recurrent reinforcement learning algorithms. Here we do the optimization on-line using a reinforcement learning technique. This reinforcement learning algorithm is based on stochastic gradient ascent. The gradient of $U_T$ with respect to the parameters $\theta$ of the system after a sequence of $T$ trades is

$$\frac{dU_T(\theta)}{d\theta} = \sum_{t=1}^{T} \frac{dU_T}{dR_t} \left\{ \frac{dR_t}{dF_t} \frac{dF_t}{d\theta} + \frac{dR_t}{dF_{t-1}} \frac{dF_{t-1}}{d\theta} \right\} \quad . \tag{6}$$

A simple on-line stochastic optimization can be obtained by considering only the term in (6) that depends on the most recently realized return $R_t$ during a forward pass through the data:

$$\frac{dU_t(\theta)}{d\theta} = \frac{dU_t}{dR_t} \left\{ \frac{dR_t}{dF_t} \frac{dF_t}{d\theta} + \frac{dR_t}{dF_{t-1}} \frac{dF_{t-1}}{d\theta} \right\} \quad . \tag{7}$$

The parameters are then updated on-line using $\Delta\theta_t = \rho dU_t(\theta_t)/d\theta_t$. Because of the recurrent structure of the problem (necessary when transaction costs are included), we use a reinforcement learning algorithm based on real-time recurrent learning (Williams & Zipser 1989). This approach, which we call recurrent reinforcement learning (RRL), is described in (Moody & Wu 1997, Moody *et al.* 1998) along with extensive simulation results.

## 4   Empirical Results: S&P 500 / TBill Asset Allocation

A long/short trading system is trained on monthly S&P 500 stock index and 3-month TBill data to maximize the differential Sharpe ratio. The S&P 500 target series is the total return index computed by reinvesting dividends. The 84 input series used in the trading systems include both financial and macroeconomic data. All data are obtained from Citibase, and the macroeconomic series are lagged by one month to reflect reporting delays.

A total of 45 years of monthly data are used, from January 1950 through December 1994. The first 20 years of data are used only for the initial training of the system. The test period is the 25 year period from January 1970 through December 1994. The experimental results for the 25 year test period are true *ex ante* simulated trading results.

For each year during 1970 through 1994, the system is trained on a moving window of the previous 20 years of data. For 1970, the system is initialized with random parameters. For the 24 subsequent years, the previously learned parameters are used to initialize the training. In this way, the system is able to adapt to changing market and economic conditions. Within the moving training window, the "RRL" systems use the first 10 years for stochastic optimization of system parameters, and the subsequent 10 years for validating early stopping of training. The networks are linear, and are regularized using quadratic weight decay during training with a

regularization parameter of 0.01. The "Qtrader" systems use a bootstrap sample of the 20 year training window for training, and the final 10 years of the training window are used for validating early stopping of training. The networks are two-layer feedforward networks with 30 tanh units in the hidden layer.

## 4.1 Experimental Results

The left panel in Figure 1 shows box plots summarizing the test performance for the full 25 year test period of the trading systems with various realizations of the initial system parameters over 30 trials for the "RRL" system, and 10 trials for the "Qtrader" system[2]. The transaction cost is set at 0.5%. Profits are reinvested during trading, and multiplicative profits are used when calculating the wealth. The notches in the box plots indicate robust estimates of the 95% confidence intervals on the hypothesis that the median is equal to the performance of the buy and hold strategy. The horizontal lines show the performance of the "RRL" voting, "Qtrader" voting and buy and hold strategies for the same test period. The annualized monthly Sharpe ratios of the buy and hold strategy, the "Qtrader" voting strategy and the "RRL" voting strategy are 0.34, 0.63 and 0.83 respectively. The Sharpe ratios calculated here are for the returns in excess of the 3-month treasury bill rate.

The right panel of Figure 1 shows results for following the strategy of taking positions based on a majority vote of the ensembles of trading systems compared with the buy and hold strategy. We can see that the trading systems go short the S&P 500 during critical periods, such as the oil price shock of 1974, the tight money periods of the early 1980's, the market correction of 1984 and the 1987 crash. This ability to take advantage of high treasury bill rates or to avoid periods of substantial stock market loss is the major factor in the long term success of these trading models. One exception is that the "RRL" trading system remains long during the 1991 stock market correction associated with the Persian Gulf war, though the "Qtrader" system does identify the correction. On the whole though, the "Qtrader" system trades much more frequently than the "RRL" system, and in the end does not perform as well on this data set.

From these results we find that both trading systems outperform the buy and hold strategy, as measured by both accumulated wealth and Sharpe ratio. These differences are statistically significant and support the proposition that there is predictability in the U.S. stock and treasury bill markets during the 25 year period 1970 through 1994. A more detailed presentation of the "RRL" results appears in (Moody *et al.* 1998).

## 4.2 Gaining Economic Insight Through Sensitivity Analysis

A sensitivity analysis of the "RRL" systems was performed in an attempt to determine on which economic factors the traders are basing their decisions. Figure 2 shows the absolute normalized sensitivities for 3 of the more salient input series as a function of time, averaged over the 30 members of the "RRL" committee. The sensitivity of input $i$ is defined as:

$$S_i = \left|\frac{dF}{dx_i}\right| \bigg/ \max_j \left|\frac{dF}{dx_j}\right| \qquad (8)$$

where $F$ is the unthresholded trading output and $x_i$ denotes input $i$.

none

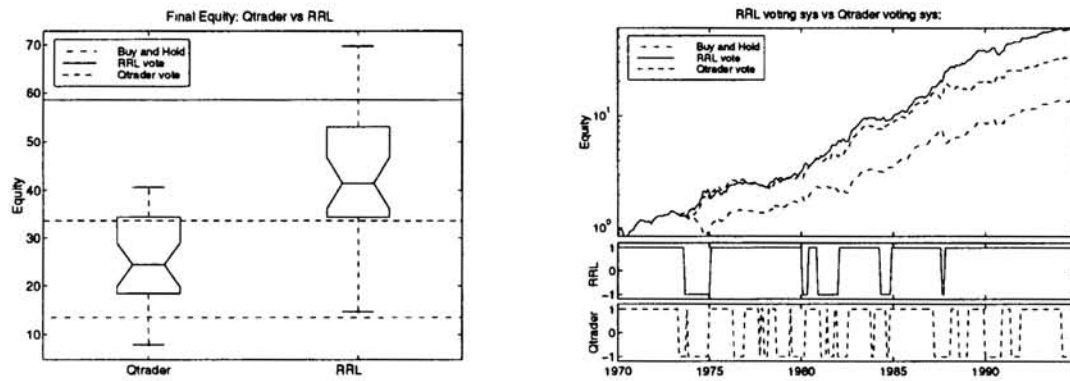

Figure 1: Test results for ensembles of simulations using the S&P 500 stock index and 3-month Treasury Bill data over the 1970-1994 time period. The solid curves correspond to the "RRL" voting system performance, dashed curves to the "Qtrader" voting system and the dashed and dotted curves indicate the buy and hold performance. The boxplots in (a) show the performance for the ensembles of "RRL" and "Qtrader" trading systems The horizontal lines indicate the performance of the voting systems and the buy and hold strategy. Both systems significantly outperform the buy and hold strategy. (b) shows the equity curves associated with the voting systems and the buy and hold strategy, as well as the voting trading signals produced by the systems. In both cases, the traders avoid the dramatic losses that the buy and hold strategy incurred during 1974 and 1987.

The time-varying sensitivities in Figure 2 emphasize the nonstationarity of economic relationships. For example, the yield curve slope (which measures inflation expectations) is found to be a very important factor in the 1970's, while trends in long term interest rates (measured by the 6 month difference in the AAA bond yield) becomes more important in the 1980's, and trends in short term interest rates (measured by the 6 month difference in the treasury bill yield) dominate in the early 1990's.

## 5    Conclusions and Extensions

In this paper, we have trained trading systems via reinforcement learning to optimize financial objective functions including our *differential Sharpe ratio* for online learning. We have also provided results that demonstrate the presence of predictability in the monthly S&P 500 Stock Index for the 25 year period 1970 through 1994.

We have previously shown with extensive simulation results (Moody & Wu 1997, Moody *et al.* 1998) that the "RRL" trading system significantly outperforms systems trained using supervised methods for traders of both single securities and portfolios. The superiority of reinforcement learning over supervised learning is most striking when state-dependent transaction costs are taken into account. Here, we present results for asset allocation systems trained using two different reinforcement learning algorithms on a real, economic dataset. We find that the "Qtrader" system does not perform as well as the "RRL" system on the S&P 500 / TBill asset allocation problem, possibly due to its more frequent trading. This effect deserves further exploration. In general, we find that Q-learning can suffer from the curse of dimensionality and is more difficult to use than our RRL approach.

Finally, we apply sensitivity analysis to the trading systems, and find that certain interest rate variables have an influential role in making asset allocation decisions.

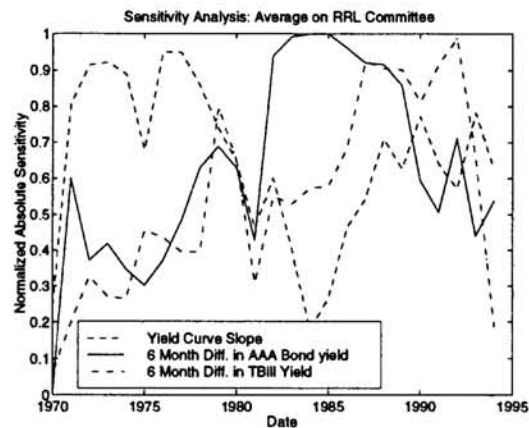

Figure 2: Sensitivity traces for three of the inputs to the "RRL" trading system averaged over the ensemble of traders. The nonstationary relationships typical among economic variables is evident from the time-varying sensitivities.

We also find that these influences exhibit nonstationarity over time.

### Acknowledgements

We gratefully acknowledge support for this work from Nonlinear Prediction Systems and from DARPA under contract DAAH01-96-C-R026 and AASERT grant DAAH04-95-1-0485.

## Footnotes

* The authors are also with Nonlinear Prediction Systems.

[1] See Moody et al. (1998) for a detailed discussion of multiple asset portfolios.

[2]Ten trials were done for the "Qtrader" system due to the amount of computation required in training the systems

## References

Crites, R. H. & Barto, A. G. (1996), Improving elevator performance using reinforcement learning, *in* D. S. Touretzky, M. C. Mozer & M. E. Hasselmo, eds, 'Advances in NIPS', Vol. 8, pp. 1017–1023.

Moody, J. & Wu, L. (1997), Optimization of trading systems and portfolios, *in* Y. Abu-Mostafa, A. N. Refenes & A. S. Weigend, eds, 'Decision Technologies for Financial Engineering', World Scientific, London, pp. 23–35. This is a slightly revised version of the original paper that appeared in the NNCM*96 Conference Record, published by Caltech, Pasadena, 1996.

Moody, J., Wu, L., Liao, Y. & Saffell, M. (1998), 'Performance functions and reinforcement learning for trading systems and portfolios', *Journal of Forecasting* 17, 441–470.

Neuneier, R. (1996), Optimal asset allocation using adaptive dynamic programming, *in* D. S. Touretzky, M. C. Mozer & M. E. Hasselmo, eds, 'Advances in NIPS', Vol. 8, pp. 952–958.

Sharpe, W. F. (1966), 'Mutual fund performance', *Journal of Business* pp. 119–138.

Tesauro, G. (1989), 'Neurogammon wins the computer olympiad', *Neural Computation* 1, 321–323.

Watkins, C. J. C. H. (1989), Learning with Delayed Rewards, PhD thesis, Cambridge University, Psychology Department.

Williams, R. J. & Zipser, D. (1989), 'A learning algorithm for continually running fully recurrent neural networks', *Neural Computation* 1, 270–280.

Zhang, W. & Dietterich, T. G. (1996), High-performance job-shop scheduling with a time-delay td($\lambda$) network, *in* D. S. Touretzky, M. C. Mozer & M. E. Hasselmo, eds, 'Advances in NIPS', Vol. 8, pp. 1024–1030.